# Message passing for task redistribution on sparse graphs

**K. Y. Michael Wong**
Hong Kong U. of Science & Technology
Clear Water Bay, Hong Kong, China
phkywong@ust.hk

**David Saad**
NCRG, Aston University
Birmingham B4 7ET, UK
D.Saad@aston.ac.uk

**Zhuo Gao**
Hong Kong U. of Science & Technology, Clear Water Bay, Hong Kong, China
Permanent address: Dept. of Physics, Beijing Normal Univ., Beijing 100875, China
zhuogao@bnu.edu.cn

## Abstract

The problem of resource allocation in sparse graphs with *real* variables is studied using methods of statistical physics. An efficient distributed algorithm is devised on the basis of insight gained from the analysis and is examined using numerical simulations, showing excellent performance and full agreement with the theoretical results.

## 1 Introduction

Optimal resource allocation is a well known problem in the area of distributed computing [1, 2] to which significant effort has been dedicated within the computer science community. The problem itself is quite general and is applicable to other areas as well where a large number of nodes are required to balance loads/resources and redistribute tasks, such as reducing internet traffic congestion [3]. The problem has many flavors and usually refers, in the computer science literature, to finding practical heuristic solutions to the distribution of computational load between computers connected in a predetermined manner.

The problem we are addressing here is more generic and is represented by nodes of some computational power that should carry out tasks. Both computational powers and tasks will be chosen at random from some arbitrary distribution. The nodes are located on a randomly chosen sparse graph of some given connectivity. The goal is to migrate tasks on the graph such that demands will be satisfied while minimizing the migration of (sub-)tasks. An important aspect of the desired algorithmic solution is that decisions on messages to be passed are carried out locally; this enables an efficient implementation of the algorithm in large non-centralized distributed networks. We focus here on the satisfiable case where the total computing power is greater than the demand, and where the number of nodes involved is very large. The unsatisfiable case can be addressed using similar techniques.

We analyze the problem using the Bethe approximation of statistical mechanics in Section 2, and alternatively a new variant of the replica method [4, 5] in Section 3. We then present numerical results in Section 4, and derive a new message passing distributed algo-

rithm on the basis of the analysis (in Section 5). We conclude the paper with a summary and a brief discussion on future work.

## 2 The statistical physics framework: Bethe approximation

We consider a typical resource allocation task on a sparse graph of $N$ nodes, labelled $i = 1, .., N$. Each node $i$ is randomly connected to $c$ other nodes[1], and has a capacity $\Lambda_i$ randomly drawn from a distribution $\rho(\Lambda_i)$. The objective is to migrate tasks between nodes such that each node will be capable of carrying out its tasks. The *current* $y_{ij} \equiv -y_{ji}$ drawn from node $j$ to $i$ is aimed at satisfying the constraint

$$\sum_j \mathcal{A}_{ij} y_{ij} + \Lambda_i \geq 0 \,, \tag{1}$$

representing the 'revised' assignment for node $i$, where $\mathcal{A}_{ij} = 1/0$ for connected/unconnected node pairs $i$ and $j$, respectively. To illustrate the statistical mechanics approach to resource allocation, we consider the load balancing task of minimizing the energy function (cost) $E = \sum_{(ij)} \mathcal{A}_{ij} \phi(y_{ij})$, where the summation $(ij)$ runs over all pairs of nodes, subject to the constraints (1); $\phi(y)$ is a general function of the current $y$. For load balancing tasks, $\phi(y)$ is typically a convex function, which will be assumed in our study. The analysis of the graph is done by introducing the free energy $F = -T \ln \mathcal{Z}_y$ for a temperature $T \equiv \beta^{-1}$, where $\mathcal{Z}_y$ is the partition function

$$\mathcal{Z}_y = \prod_{(ij)} \int dy_{ij} \prod_i \Theta \left( \sum_j \mathcal{A}_{ij} y_{ij} + \Lambda_i \right) \exp \left[ -\beta \sum_{(ij)} \mathcal{A}_{ij} \phi(y_{ij}) \right] . \tag{2}$$

The $\Theta$ function returns 1 for a non-negative argument and 0 otherwise.

When the connectivity $c$ is low, the probability of finding a loop of finite length on the graph is low, and the Bethe approximation well describes the local environment of a node. In the approximation, a node is connected to $c$ branches in a tree structure, and the correlations among the branches of the tree are neglected. In each branch, nodes are arranged in generations. A node is connected to an ancestor node of the previous generation, and another $c - 1$ descendent nodes of the next generation.

Consider a vertex $V(\mathbf{T})$ of capacity $\Lambda_{V(\mathbf{T})}$, and a current $y$ is drawn from the vertex. One can write an expression for the free energy $F(y|\mathbf{T})$ as a function of the free energies $F(y_k|\mathbf{T}_k)$ of its descendants, that branch out from this vertex

$$F(y|\mathbf{T}) = -T \ln \left\{ \prod_{k=1}^{c-1} \left( \int dy_k \right) \Theta \left( \sum_{k=1}^{c-1} y_k - y + \Lambda_V \right) \right.$$
$$\left. \times \exp \left[ -\beta \sum_{k=1}^{c-1} (F(y_k|\mathbf{T}_k) + \phi(y_k)) \right] \right\}, \tag{3}$$

where $\mathbf{T}_k$ represents the tree terminated at the $k^{\text{th}}$ descendent of the vertex. The free energy can be considered as the sum of two parts, $F(y|\mathbf{T}) = N_{\mathbf{T}} F_{\text{av}} + F_V(y|\mathbf{T})$, where $N_{\mathbf{T}}$ is the number of nodes in the tree $\mathbf{T}$, $F_{\text{av}}$ is the average free energy per node, and $F_V(y|\mathbf{T})$ is referred to as the *vertex free energy*[2]. Note that when a vertex is added to a tree, there is a

change in the free energy due to the added vertex. Since the number of nodes increases by 1, the vertex free energy is obtained by subtracting the free energy change by the average free energy. This allows us to obtain the recursion relation

$$F_V(y|\mathbf{T}) = -T \ln \left\{ \prod_{k=1}^{c-1} \left( \int dy_k \right) \Theta \left( \sum_{k=1}^{c-1} y_k - y + \Lambda_{V(\mathbf{T})} \right) \right.$$
$$\left. \times \exp \left[ -\beta \sum_{k=1}^{c-1} \left( F_V(y_k|\mathbf{T}_k) + \phi(y_k) \right) \right] \right\} - F_{\text{av}}, \qquad (4)$$

and the average free energy per node is given by

$$F_{\text{av}} = -T \left\langle \ln \left\{ \prod_{k=1}^{c} \left( \int dy_k \right) \Theta \left( \sum_{k=1}^{c} y_k + \Lambda_V \right) \right. \right.$$
$$\left. \left. \times \exp \left[ -\beta \sum_{k=1}^{c} \left( F_V(y_k|\mathbf{T}_k) + \phi(y_k) \right) \right] \right\} \right\rangle_\Lambda, \qquad (5)$$

where $\Lambda_V$ is the capacity of the vertex $V$ fed by $c$ trees $\mathbf{T}_1, \ldots, \mathbf{T}_c$, and $\langle \ldots \rangle_\Lambda$ represents the average over the distribution $\rho(\Lambda)$. In the zero temperature limit, Eq. (4) reduces to

$$F_V(y|\mathbf{T}) = \min_{\{y_k \,|\, \sum_{k=1}^{c-1} y_k - y + \Lambda_{V(\mathbf{T})} \geq 0\}} \left[ \sum_{k=1}^{c-1} \left( F_V(y_k|\mathbf{T}_k) + \phi(y_k) \right) \right] - F_{\text{av}}. \qquad (6)$$

The current distribution and the average free energy per link can be derived by integrating the current $y'$ in a link from one vertex to another, fed by the trees $\mathbf{T}_1$ and $\mathbf{T}_2$, respectively; the obtained expressions are $P(y) = \langle \delta(y - y') \rangle_\star$ and $\langle E \rangle = \langle \phi(y') \rangle_\star$ where

$$\langle \bullet \rangle_\star = \left\langle \frac{\int dy' \exp \left[ -\beta \left( F_V(y'|\mathbf{T}_1) + F_V(-y'|\mathbf{T}_2) + \phi(y') \right) \right] (\bullet)}{\int dy' \exp \left[ -\beta \left( F_V(y'|\mathbf{T}_1) + F_V(-y'|\mathbf{T}_2) + \phi(y') \right) \right]} \right\rangle_\Lambda. \qquad (7)$$

## 3  The statistical physics framework: replica method

In this section, we sketch the analysis of the problem using the replica method, as an alternative to the Bethe approximation. The derivation is rathe involved, details will be provided elsewhere. To facilitate derivations, we focus on the quadratic cost function $\phi(y) = y^2/2$. The results confirm the validity of the Bethe approximation on sparse graphs.

An alternative formulation of the original optimization problem is to consider its dual. Introducing Lagrange multipliers, the function to be minimized becomes $L = \sum_{(ij)} \mathcal{A}_{ij} y_{ij}^2/2 + \sum_i \mu_i(\sum_j \mathcal{A}_{ij} y_{ij} + \Lambda_i)$. Optimizing $L$ with respect to $y_{ij}$, one obtains $y_{ij} = \mu_j - \mu_i$, where $\mu_i$ is referred to as the *chemical potential* of node $i$, and the current is driven by the potential difference.

Although the analysis has also been carried out in the space of currents, we focus here on the optimization problem in the space of the chemical potentials. Since the energy function is invariant under the addition of an arbitrary global constant to the chemical potentials of all nodes, we introduce an extra regularization term $\epsilon \sum_i \mu_i^2/2$ to break the translational symmetry, where $\epsilon \to 0$. To study the characteristics of the problem one calculates the averaged free energy per node $F_{\text{av}} = -T \langle \ln \mathcal{Z}_\mu \rangle_{A,\Lambda}/N$, where $\mathcal{Z}_\mu$ is the partition function

$$\prod_i \left[ \int d\mu_i \, \Theta \left( \sum_j \mathcal{A}_{ij}(\mu_j - \mu_i) + \Lambda_i \right) \right] \exp \left[ -\frac{\beta}{2} \left( \sum_{(ij)} \mathcal{A}_{ij}(\mu_j - \mu_i)^2 + \epsilon \sum_i \mu_i^2 \right) \right].$$

The calculation follows the main steps of a replica based calculation in diluted systems [6], using the identity $\ln \mathcal{Z} = \lim_{n\to 0}[\mathcal{Z}^n - 1]/n$. The replicated partition function [5] is averaged over all network configurations with connectivity and capacity distributions $\rho(\Lambda_i)$. We consider the case of intensive connectivity $c \sim O(1) \ll N$. Extending the analysis of [6] and averaging over all connectivity matrices, one finds

$$
\langle \mathcal{Z}_\mu^n \rangle = \quad \exp N \left\{ \frac{c}{2} - c \sum_{\mathbf{r,s}} \hat{Q}_{\mathbf{r,s}} Q_{\mathbf{r,s}} + \ln \int d\Lambda \rho(\Lambda) \prod_\alpha \left( \int d\mu_\alpha \int_\Lambda^\infty d\lambda_\alpha \int \frac{d\hat{\lambda}_\alpha}{2\pi} \right) \right.
$$
$$
\left. \times \exp \left[ \sum_\alpha \left( i\hat{\lambda}_\alpha (\lambda_\alpha + c\mu_\alpha) - \frac{\beta}{2}(c+\epsilon)\mu_\alpha^2 \right) \right] X^c \right\}, \tag{8}
$$

where $X = \sum_{\mathbf{r,s}} \hat{Q}_{\mathbf{r,s}} \prod_\alpha (-i\hat{\lambda}_\alpha)^{r_\alpha} \mu_\alpha^{s_\alpha} + \sum_{\mathbf{r,s}} \frac{Q_{\mathbf{r,s}}}{2 \prod_\alpha r_\alpha! s_\alpha!} \prod_\alpha \mu_\alpha^{r_\alpha} (\beta\mu_\alpha - i\hat{\lambda}_\alpha)^{s_\alpha}$. The order parameters $Q_{\mathbf{r,s}}$ and $\hat{Q}_{\mathbf{r,s}}$, are labelled by the somewhat unusual indices $\mathbf{r}$ and $\mathbf{s}$, representing the $n$-component integer vectors $(r_1, .., r_n)$ and $(s_1, .., s_n)$ respectively. This is a result of the specific interaction considered which entangles nodes of different indices. The order parameters $Q_{\mathbf{r,s}}$ and $\hat{Q}_{\mathbf{r,s}}$ are given by the extremum condition of Eq. (8), i.e., via a set of saddle point equations w.r.t the order parameters. Assuming replica symmetry, the saddle point equations yield a recursion relation for a two-component function $R$, which is related to the order parameters via the generating function

$$
P_{\mathbf{s}}(\mathbf{z}) = \sum_{\mathbf{r}} Q_{\mathbf{r,s}} \prod_\alpha \frac{(z_\alpha)^{r_\alpha}}{r_\alpha!} = \left\langle \prod_\alpha \left( \int d\mu\, R(z_\alpha, \mu | \mathbf{T}) e^{-\beta\mu^2/2} \mu^{s_\alpha} \right) \right\rangle_\Lambda . \tag{9}
$$

In Eq. (9), $\mathbf{T}$ represents the tree terminated at the vertex node with chemical potential $\mu$, providing input to the ancestor node with chemical potential $z$, and $\langle \ldots \rangle_\Lambda$ represents the average over the distribution $\rho(\Lambda)$. The resultant recursion relation for $R(z, \mu | \mathbf{T})$ is independent of the replica indices, and is given by

$$
R(z, \mu | \mathbf{T}) = \quad \frac{1}{\mathcal{D}} \prod_{k=1}^{c-1} \left( \int d\mu_k R(\mu, \mu_k | \mathbf{T}_k) \right) \Theta \left( \sum_{k=1}^{c-1} \mu_k - c\mu + z + \Lambda_{V(\mathbf{T})} \right)
$$
$$
\times \exp \left[ -\frac{\beta}{2} \left( \sum_{k=1}^{c-1} (\mu - \mu_k)^2 + \epsilon\mu^2 \right) \right], \tag{10}
$$

where the vertex node has a capacity $\Lambda_{V(\mathbf{T})}$; $\mathcal{D}$ is a constant. $R(z, \mu | \mathbf{T})$ is expressed in terms of $c-1$ functions $R(\mu, \mu_k | \mathbf{T}_k)$ ($k = 1, .., c-1$), integrated over $\mu_k$. This algebraic structure is typical of the Bethe lattice tree-like representation of networks of connectivity $c$, where a node obtains input from its $c-1$ descendent nodes of the next generation, and $\mathbf{T}_k$ represents the tree terminated at the $k^{\text{th}}$ descendent.

Except for the regularization factor $\exp(-\beta\epsilon\mu^2/2)$, $R$ turns out to be a function of $y \equiv \mu - z$, which is interpreted as the current drawn from a node with chemical potential $\mu$ by its ancestor with chemical potential $z$. One can then express the function $R$ as the product of a *vertex partition function* $Z_V$ and a normalization factor $W$, that is, $R(z, \mu | \mathbf{T}) = W(\mu) Z_V(y | \mathbf{T})$. In the limit $n \to 0$, the dependence on $\mu$ and $y$ are separable, providing a recursion relation for $Z_V(y | \mathbf{T})$. This gives rise to the *vertex free energy* $F_V(y | \mathbf{T}) = -T \ln Z_V(y | \mathbf{T})$ when a current $y$ is drawn from the vertex of a tree $\mathbf{T}$. The recursive equation and the average free energy expression agrees with the results in the Bethe approximation. These iterative equations can be directly linked to those obtained from a principled Bayesian approximation, where the logarithms of the messages passed between nodes are proportional to the vertex free energies.

## 4 Numerical solution

The solution of Eq. (6) is obtained numerically. Since the vertex free energy of a node depends on its own capacity and the disordered configuration of its descendants, we generate 1000 nodes at each iteration of Eq. (6), with capacities randomly drawn from the distribution $\rho(\Lambda)$, each being fed by $c-1$ nodes randomly drawn from the previous iteration.

We have discretized the vertex free energies $F_V(y|\mathbf{T})$ function into a vector, whose $i^{\text{th}}$ component is the value of the function corresponding to the current $y_i$. To speed up the optimization search at each node, we first find the *vertex saturation current* drawn from a node such that: (a) the capacity of the node is just used up; (b) the current drawn by each of its descendant nodes is just enough to saturate its own capacity constraint. When these conditions are satisfied, we can separately optimize the current drawn by each descendant node, and the vertex saturation current is equal to the node capacity subtracted by the current drawn by its descendants. The optimal solution can be found using an exhaustive search, by varying the component currents in small discrete steps. This approach is particularly convenient for $c = 3$, where the search is confined to a single parameter.

To compute the average energy, we randomly draw 2 nodes, compute the optimal current flowing between them, and repeat the sampling to obtain the average. Figure 1(a) shows the results as a function of iteration step $t$, for a Gaussian capacity distribution $\rho(\Lambda)$ with variance 1 and average $\langle\Lambda\rangle$. Each iteration corresponds to adding one extra generation to the tree structure, such that the iterative process corresponds to approximating the network by an increasingly extensive tree. We observe that after an initial rise with iteration steps, the average energies converges to steady-state values, at a rate which increases with the average capacity.

To study the convergence rate of the iterations, we fit the average energy at iteration step $t$ using $\langle E(t) - E(\infty)\rangle \sim \exp(-\gamma t)$ in the asymptotic regime. As shown in the inset of Fig. 1(a), the relaxation rate $\gamma$ increases with the average capacity. It is interesting to note that a cusp exists at the average capacity of about 0.45. Below that value, convergence of the iteration is slow, since the average energy curve starts to develop a plateau before the final convergence. On the other hand, the plateau disappears and the convergence is fast above the cusp. The slowdown of convergence below the cusp is probably due to the appearance of increasingly large clusters of nonzero currents on the network, since clusters of nodes with negative capacities become increasingly extensive, and need to draw currents from increasingly extensive regions of nodes with excess capacities to satisfy the demand. Figure 1(b) illustrates the current distribution for various average capacities. The distribution $P(y)$ consists of a delta function component at $y = 0$ and a continuous component whose breadth decreases with average capacity. The fraction of links with zero currents increases with the average capacity. Hence at a low average capacity, links with nonzero currents form a percolating cluster, whereas at a high average capacity, it breaks into isolated clusters.

## 5 Distributed algorithms

The local nature of the recursion relation Eq. (6) points to the possibility that the network optimization can be solved by message passing approaches, which have been successful in problems such as error-correcting codes [8] and probabilistic inference [9]. The major advantage of message passing is its potential to solve a global optimization problem via local updates, thereby reducing the computational complexity. For example, the computational complexity of quadratic programming for the load balancing task typically scales as $N^3$, whereas capitalizing on the network topology underlying the connectivity of the variables, message passing scales as $N$. An even more important advantage, relevant to

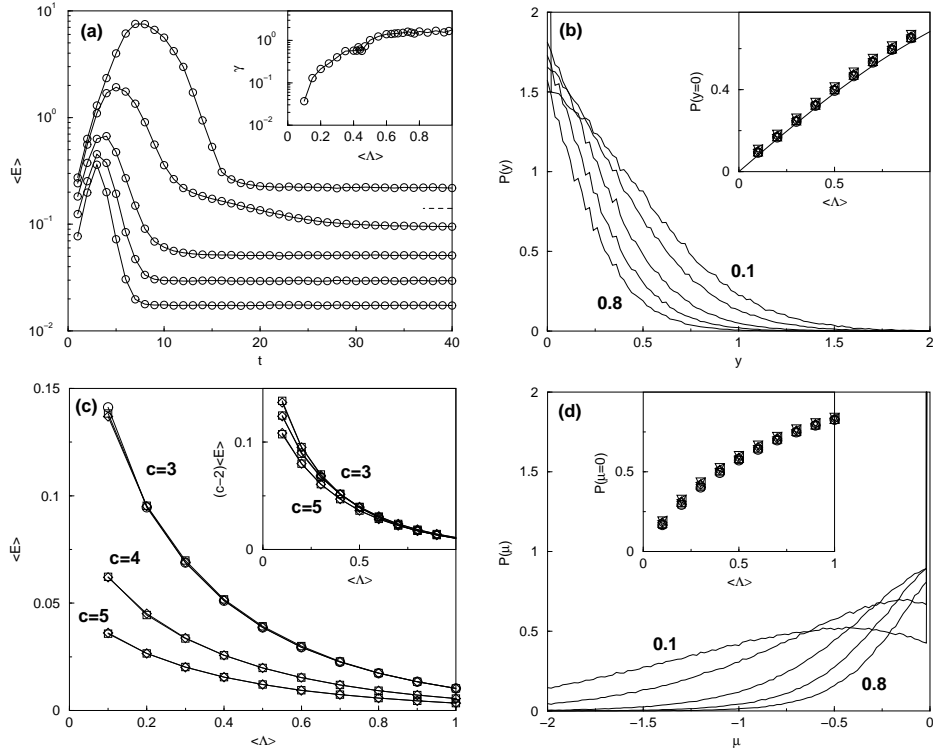

Figure 1: Results for system size $N = 1000$ and $\phi(y) = y^2/2$. (a) $\langle E \rangle$ obtained by iterating Eq. (6) as a function of $t$ for $\langle \Lambda \rangle = 0.1, 0.2, 0.4, 0.6, 0.8$ (top to bottom) and $c = 3$. Dashed line: The asymptotic $\langle E \rangle$ for $\langle \Lambda \rangle = 0.1$. Inset: $\gamma$ as a function of $\langle \Lambda \rangle$. (b) The distribution $P(y)$ obtained by iterating Eq. (6) to steady states for the same parameters and average capacities as in (a), from right to left. Inset: $P(y = 0)$ as a function of $\langle \Lambda \rangle$. Symbols: $c = 3$ ($\bigcirc$) and ($\square$), $c = 4$ ($\lozenge$) and ($\triangle$), $c = 5$ ($\triangleleft$) and ($\nabla$); each pair obtained from Eqs. (11) and (14) respectively. Line: $\mathrm{erf}(\langle \Lambda \rangle / \sqrt{2})$. (c) $\langle E \rangle$ as a function of $\langle \Lambda \rangle$ for $c = 3, 4, 5$. Symbols: results of Eq. (6) ($\bigcirc$), Eq.(11) ($\square$), and Eq. (14) ($\lozenge$). Inset: $\langle E \rangle$ multiplied by $(c - 2)$ as a function of $\langle \Lambda \rangle$ for the same conditions. (d) The distribution $P(\mu)$ obtained by iterating Eq. (14) to steady states for the same parameters and average capacities as in (b), from left to right. Inset: $P(\mu = 0)$ as a function of $\langle \Lambda \rangle$. Symbols: same as (b).

practical implementation, is its distributive nature; it does not require a global optimizer, and is particularly suitable for distributive control in evolving networks.

However, in contrast to other message passing algorithms which pass conditional probability estimates of *discrete variables* to neighboring nodes, the messages in the present context are more complex, since they are *functions* $F_V(y|\mathbf{T})$ of the current $y$. We simplify the message to 2 parameters, namely, the first and second derivatives of the vertex free energies. For the quadratic load balancing task, it can be shown that a self-consistent solution of the recursion relation, Eq. (6), consists of vertex free energies which are piecewise quadratic with continuous slopes. This makes the 2-parameter message a very precise approximation.

Let $(A_{ij}, B_{ij}) \equiv (\partial F_V(y_{ij}|\mathbf{T}_j)/\partial y_{ij}, \partial^2 F_V(y_{ij}|\mathbf{T}_j)/\partial y_{ij}^2)$ be the message passed from

node $j$ to $i$; using Eq.(6), the recursion relation of the messages become

$$A_{ij} \leftarrow -\mu_{ij}, \quad B_{ij} \leftarrow \Theta(-\mu_{ij}) \left[ \sum_{k \neq i} \mathcal{A}_{jk} (\phi''_{jk} + B_{jk})^{-1} \right]^{-1}, \quad \text{where} \qquad (11)$$

$$\mu_{ij} = \min \left[ \frac{\sum_{k \neq i} \mathcal{A}_{jk} [y_{jk} - (\phi'_{jk} + A_{jk})(\phi''_{jk} + B_{jk})^{-1}] + \Lambda_j - y_{ij}}{\sum_{k \neq i} \mathcal{A}_{jk} (\phi''_{jk} + B_{jk})^{-1}}, 0 \right], \qquad (12)$$

with $\phi'_{jk}$ and $\phi''_{jk}$ representing the first and second derivatives of $\phi(y)$ at $y = y_{jk}$ respectively. The forward passing of the message from node $j$ to $i$ is then followed by a backward message from node $j$ to $k$ for updating the currents $y_{jk}$ according to

$$y_{jk} \leftarrow y_{jk} - \frac{\phi'_{jk} + A_{jk} + \mu_{ij}}{\phi''_{jk} + B_{jk}}. \qquad (13)$$

We simulate networks with $c = 3$, $\phi(y) = y^2/2$ and compute their average energies. The network configurations are generated randomly, with loops of lengths 3 or less excluded. Updates are performed with random sequential choices of the nodes. As shown in Fig. 1(c), the simulation results of the message passing algorithm have an excellent agreement with those obtained by the recursion relation Eq.(6).

For the quadratic load balancing task considered here, an independent exact optimization is available for comparison. The Kühn-Tucker conditions for the optimal solution yields

$$\mu_i = \min \left[ \frac{1}{c} \left( \sum_j \mathcal{A}_{ij} \mu_j + \Lambda_i \right), 0 \right]. \qquad (14)$$

It also provides a local iterative method for the optimization problem. As shown in Fig. 1(c), both the recursion relation Eq.(6) and the message passing algorithm Eq.(11) yield excellent agreement with the iteration of chemical potentials Eq.(14).

Both Eqs. (11) and (14) allow us to study the distribution $P(\mu)$ of the chemical potentials $\mu$. As shown in Fig. 1(d), $P(\mu)$ consists of a delta function and a continuous component. Nodes with zero chemical potentials correspond to those with unsaturated capacity constraints. The fraction of unsaturated nodes increases with the average capacity, as shown in the inset of Fig. 1(d). Hence at a low average capacity, saturated nodes form a percolating cluster, whereas at a high average capacity, it breaks into isolated clusters. It is interesting to note that at the average capacity of 0.45, below which a plateau starts to develop in the relaxation rate of the recursion relation Eq. (6), the fraction of unsaturated nodes is about 0.53, close to the percolation threshold of 0.5 for $c = 3$.

Besides the case of $c = 3$, Fig. 1(c) also shows the simulation results of the average energy for $c = 4, 5$, using both Eqs. (11) and (14). We see that the average energy decreases when the connectivity increases. This is because the increase in links connecting a node provides more freedom to allocate resources. When the average capacity is 0.2 or above, an exponential fit $\langle E \rangle \sim \exp(-k\langle \Lambda \rangle)$ is applicable, where $k$ lies in the range 2.5 to 2.7. Remarkably, multiplying by a factor of $(c-2)$, we find that the 3 curves collapse in this regime of average capacity, showing that the average energy scales as $(c-2)^{-1}$ in this regime, as shown in the inset of Fig. 1(c).

Further properties of the optimized networks have been studied by simulations, and will be presented elsewhere. Here we merely summarize the main results. (a) When the average capacity drops below 0.1, the energy rises above the exponential fit applicable to the average capacity above 0.2. (b) The fraction of links with zero currents increases with the average capacity, and is rather insensitive to the connectivity. Remarkably, except for

very small average capacities, the function $\mathrm{erf}\left(\langle\Lambda\rangle/\sqrt{2}\right)$ has a very good fit with the data. Indeed, in the limit of large $\langle\Lambda\rangle$, this function approaches the fraction of links with both vertices unsaturated, that is, $[\int_0^\infty d\Lambda\rho(\Lambda)]^2$. (c) The fraction of unsaturated nodes increases with the average capacity, and is rather insensitive to the connectivity. In the limit of large average capacities, it approaches the upper bound of $\int_0^\infty d\Lambda\rho(\Lambda)$, which is the probability that the capacity of a node is non-negative. (d) The convergence time of Eq. (11) can be measured by the time for the r.m.s. of the changes in the chemical potentials to fall below a threshold. Similarly, the convergence time of Eq. (14) can be measured by the time for the r.m.s. of the sums of the currents in both message directions of a link to fall below a threshold. When the average capacity is 0.2 or above, we find the power-law dependence on the average capacity, the exponent ranging from $-1$ for $c = 3$ to $-0.8$ for $c = 5$ for Eq. (14), and being about -0.5 for $c = 3, 4, 5$ for Eq. (11). When the average capacity decreases further, the convergence time deviates above the power laws.

## 6   Summary

We have studied a prototype problem of resource allocation on sparsely connected networks using the replica method, resulting in recursion relations interpretable using the Bethe approximation. The resultant recursion relation leads to a message passing algorithm for optimizing the average energy, which significantly reduces the computational complexity of the global optimization task and is suitable for online distributive control. The suggested 2-parameter approximation produces results with excellent agreement with the original recursion relation. For the simple but illustrative example in this letter, we have considered a quadratic cost function, resulting in an exact algorithm based on local iterations of chemical potentials, and the message passing algorithm shows remarkable agreement with the exact result. The suggested simple message passing algorithm can be generalized to more realistic cases of nonlinear cost functions and additional constraints on the capacities of nodes and links. This constitutes a rich area for further investigations with many potential applications.

### Acknowledgments

This work is partially supported by research grants HKUST6062/02P and DAG04/05.SC25 of the Research Grant Council of Hong Kong and by EVERGROW, IP No. 1935 in the FET, EU FP6 and STIPCO EU FP5 contract HPRN-CT-2002-00319.

## Footnotes

[1]Although we focus here on graphs of fixed connectivity, one can easily accommodate any connectivity profile within the same framework; the algorithms presented later are completely general.

[2]This term is marginalized over all inputs to the current vertex, leaving the difference in chemical potential $y$ as its sole argument, hence the terminology used.

## References

[1] Peterson L. and Davie B.S., *Computer Networks: A Systems Approach*, Academic Press, San Diego CA (2000)

[2] Ho Y.C., Servi L. and Suri R. *Large Scale Systems* **1** (1980) 51

[3] Shenker S., Clark D., Estrin D. and Herzog S. *ACM Computer Comm. Review* **26** (1996) 19

[4] Nishimori H. *Statistical Physics of Spin Glasses and Information Processing*, OUP UK (2001)

[5] Mézard M., Parisi P. and Virasoro M., *Spin Glass Theory and Beyond*, World Scientific, Singapore (1987)

[6] Wong K.Y.M. and Sherrington D. *J. Phys. A* **20**(1987) L793

[7] Sherrington D. and Kirkpatrick S. *Phys. Rev. Lett.* **35** (1975) 1792

[8] Opper M. and Saad D. *Advanced Mean Field Methods*, MIT press (2001)

[9] MacKay D.J.C., *Information Theory, Inference and Learning Algorithms*, CUP UK(2003)
